# A DYNAMICAL APPROACH TO TEMPORAL PATTERN PROCESSING

W. Scott Stornetta
Stanford University, Physics Department, Stanford, Ca., 94305

Tad Hogg and B. A. Huberman
Xerox Palo Alto Research Center, Palo Alto, Ca. 94304

## ABSTRACT

Recognizing patterns with temporal context is important for such tasks as speech recognition, motion detection and signature verification. We propose an architecture in which time serves as its own representation, and temporal context is encoded in the state of the nodes. We contrast this with the approach of replicating portions of the architecture to represent time.

As one example of these ideas, we demonstrate an architecture with capacitive inputs serving as temporal feature detectors in an otherwise standard back propagation model. Experiments involving motion detection and word discrimination serve to illustrate novel features of the system. Finally, we discuss possible extensions of the architecture.

## INTRODUCTION

Recent interest in connectionist, or "neural" networks has emphasized their ability to store, retrieve and process patterns[1,2]. For most applications, the patterns to be processed are static in the sense that they lack temporal context.

Another important class consists of those problems that require the processing of temporal patterns. In these the information to be learned or processed is not a particular pattern but a sequence of patterns. Such problems include speech processing, signature verification, motion detection, and predictive signal processing[3-8].

More precisely, temporal pattern processing means that the desired output depends not only on the current input but also on those preceding or following it as well. This implies that two identical inputs at different time steps might yield different desired outputs depending on what patterns precede or follow them.

There is another feature characteristic of much temporal pattern processing. Here an entire sequence of patterns is recognized as a single distinct category,

generating a single output. A typical example of this would be the need to recognize words from a rapidly sampled acoustic signal. One should respond only once to the appearance of each word, even though the word consists of many samples. Thus, each input may not produce an output.

With these features in mind, there are at least three additional issues which networks that process temporal patterns must address, above and beyond those that work with static patterns. The first is how to represent temporal context in the state of the network. The second is how to train at intermediate time steps before a temporal pattern is complete. The third issue is how to interpret the outputs during recognition, that is, how to tell when the sequence has been completed. Solutions to each of these issues require the construction of appropriate input and output representations. This paper is an attempt to address these issues, particularly the issue of representing temporal context in the state of the machine. We note in passing that the recognition of temporal sequences is distinct from the related problem of generating a sequence, given its first few members[9,10,11].

## TEMPORAL CLASSIFICATION

With some exceptions[10,12], in most previous work on temporal problems the systems record the temporal pattern by replicating part of the architecture for each time step. In some instances input nodes and their associated links are replicated[3,4]. In other cases only the weights or links are replicated, once for each of several time delays[7,8]. In either case, this amounts to mapping the temporal pattern into a spatial one of much higher dimension before processing.

These systems have generated significant and encouraging results. However, these approaches also have inherent drawbacks. First, by replicating portions of the architecture for each time step the amount of redundant computation is significantly increased. This problem becomes extreme when the signal is sampled very frequently[4]. Next, by relying on replications of the architecture for each time step, the system is quite inflexible to variations in the rate at which the data is presented or size of the temporal window. Any variability in the rate of the input signal can generate an input pattern which bears little or no resemblance to the trained pattern. Such variability is an important issue, for example, in speech recognition. Moreover, having a temporal window of any fixed length makes it manifestly impossible to detect contextual effects on time scales longer than the window size. An additional difficulty is that a misaligned signal, in its spatial representation, may have very little resemblance to the correctly aligned training signal. That is, these systems typically suffer from not being translationally invariant in time.

Networks based on relaxation to equilibrium[11,13,14] also have difficulties for use with temporal problems. Such an approach removes any dependence on initial conditions and hence is difficult to reconcile directly with temporal problems, which by their nature depend on inputs from earlier times. Also, if a temporal problem is to be handled in terms of relaxation to equilibrium, the equilibrium points themselves must be changing in time.

## A NON-REPLICATED, DYNAMIC ARCHITECTURE

We believe that many of the difficulties mentioned above are tied to the attempt to map an inherently dynamical problem into a static problem of higher dimension. As an alternative, we propose to represent the history of the inputs in the state of the nodes of a system, rather than by adding additional units. Such an approach to capturing temporal context shows some very immediate advantages over the systems mentioned above. First, it requires no replication of units for each distinct time step. Second, it does not fix in the architecture itself the window for temporal context or the presentation rate. These advantages are a direct result of the decision to let time serve as its own representation for temporal sequences, rather than creating additional spatial dimensions to represent time.

In addition to providing a solution to the above problems, this system lends itself naturally to interpretation as an evolving dynamical system. Our approach allows one to think of the process of mapping an evolving input into a discrete sequence of outputs (such as mapping continuous speech input into a sequence of words) as a dynamical system moving from one attractor to another[15].

As a preliminary example of the application of these ideas, we introduce a system that captures the temporal context of input patterns without replicating units for each time step. We modify the conventional back propagation algorithm by making the input units capacitive. In contrast to the conventional architecture in which the input nodes are used simply to distribute the signal to the next layer, our system performs an additional computation. Specifically, let $X_i$ be the value computed by an input node at time $t_i$, and $I_i$ be the input signal to this node at the same time. Then the node computes successive values according to

$$X_{i+1} = aI_{i+1} + dX_i \qquad (1)$$

where $a$ is an input amplitude and $d$ is a decay rate. Thus, the result computed by an input unit is the sum of the current input value multiplied by $a$, plus a fractional part, $d$, of the previously computed value of the input unit. In the absence of further input, this produces an exponential decay in the activation of the input nodes. The value for $d$ is chosen so that this decay reaches $1/e$ of its original value in a time $\tau$ characteristic of the time scale for the particular problem, i.e., $d = e^{-tr}$, where $r$ is the presentation rate. The value for $a$ is chosen to produce a specified maximum value for X, given by

$aI_{max}/(1-d)$. We note that Eq. (1) is equivalent to having a non-modifiable recurrent link with weight $d$ on the input nodes, as illustrated in Fig. 1.

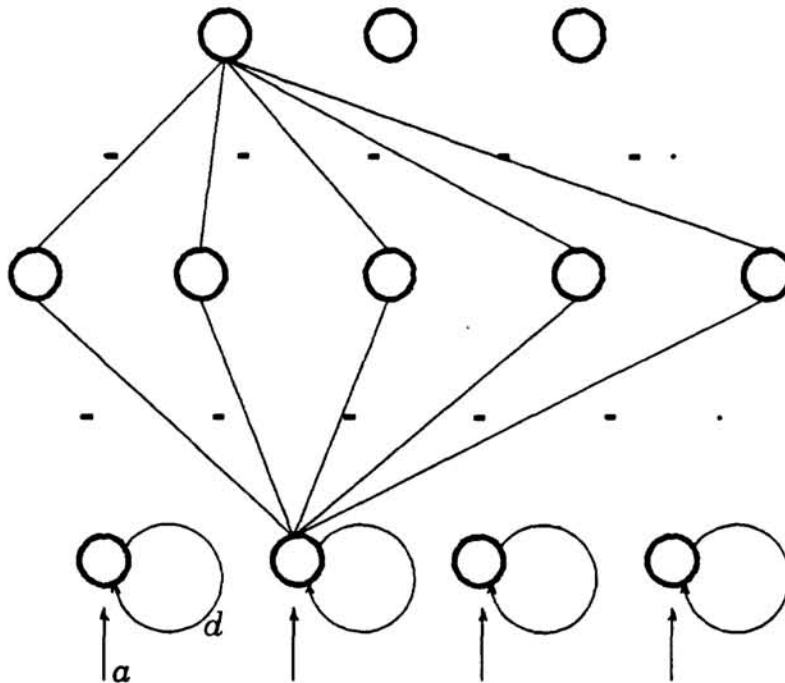

Fig. 1: Schematic architecture with capacitive inputs. The input nodes compute values according to Eq. (1). Hidden and output units are identical to standard back propagation nets.

The processing which takes place at the input node can also be thought of in terms of an infinite impulse response (IIR) digital filter. The infinite impulse response of the filter allows input from the arbitrarily distant past to influence the current output of the filter, in contrast to methods which employ fixed windows, which can be viewed in terms of finite impulse response (FIR) filters. The capacitive node of Fig. 1 is equivalent to pre-processing the signal with a filter with transfer function $a/(1-dz^{-1})$.

This system has the unique feature that a simple transformation of the parameters $a$ and $d$ allows it to respond in a near-optimal way to a signal which differs from the training signal in its rate. Consider a system initially trained at rate $r$ with decay rate $d$ and amplitude $a$. To make use of these weights for a different presentation rate, $r'$, one simply adjusts the values $a'$ and $d'$ according to

$$d' = d^{r/r'} \tag{2}$$

$$a' = a \frac{1-d'}{1-d} \tag{3}$$

These equations can be derived by the following argument. The general idea is that the values computed by the input nodes at the new rate should be as close as possible to those computed at the original rate. Specifically, suppose one wishes to change the sampling rate from $r$ to $nr$, where $n$ is an integer. Suppose that at a time $t_0$ the computed value of the input node is $X_0$. If this node receives no additional input, then after $m$ time steps, the computed value of the input node will be $X_0 d^m$. For the more rapid sampling rate, $X_0 d^m$ should be the value obtained after $nm$ time steps. Thus we require

$$X_0 d^m = X_0 d'^{mn} \tag{4}$$

which leads to Eq. (2) because $n = r'/r$. Now suppose that an input I is presented $m$ times in succession to an input node that is initially zero. After the $m^{th}$ presentation, the computed value of the input node is

$$aI \frac{1-d^m}{1-d} \tag{5}$$

Requiring this value to be equal to the corresponding value for the faster presentation rate after $nm$ time steps leads to Eq. (3). These equations, then, make the computed values of the input nodes identical, independent of the presentation rate. Of course, this statement only holds exactly in the limit that the computed values of the input nodes change only infinitesimally from one time step to the next. Thus, in practice, one must insure that the signal is sampled frequently enough that the computed value of the input nodes is slowly changing.

The point in weight space obtained after initial training at the rate $r$ has two desirable properties. First, it can be trained on a signal at one sampling rate and then the values of the weights arrived at can be used as a near-optimal starting point to further train the system on the same signal but at a different sampling rate. Alternatively, the system can respond to temporal patterns which differ in rate from the training signal, without any retraining of the weights. These factors are a result of the choice of input representation, which essentially present the same pattern to the hidden unit and other layers, independent of sampling rate. These features highlight the fact that in this system the weights to some degree represent the temporal pattern independent of the rate of presentation. In contrast, in systems which use temporal windows, the weights obtained after training on a signal at one sampling rate would have little or no relation to the desired values of the weights for a different sampling rate or window size.

## EXPERIMENTS

As an illustration of this architecture and related algorithm, a three-layer, 15-30-2 system was trained to detect the leftward or rightward motion of a gaussian pulse moving across the field of input units with sudden changes in direction. The values of $d$ and $a$ were 0.7788 and 0.4424, respectively. These values were chosen to give a characteristic decay time of 4 time steps with a maximum value computed by the input nodes of 2.0. The pulse was of unit height with a half-width, $\sigma$, of 1.3. Figure 2 shows the input pulse as well as the values computed by the input nodes for leftward or rightward motion. Once trained at a velocity of 0.1 unit per sampling time, the velocity was varied over a wide range, from a factor of 2 slower to a factor of 2 faster as shown in Fig. 3. For small variations in velocity the system continued to correctly identify the type of motion. More impressive was its performance when the scaling relations given in Eqs. (2) and (3) were used to modify the amplitude and decay rate. In this case, acceptable performance was achieved over the entire range of velocities tested. This was without any additional retraining at the new rates. The difference in performance between the two curves also demonstrates that the excellent performance of the system is not an anomaly of the particular problem chosen, but characteristic of rescaling $a$ and $d$ according to Eqs. (2) and (3). We thus see that a simple use of capacitive links to store temporal context allows for motion detection at variable velocities.

A second experiment involving speech data was performed to compare the system's performance to the time-delay-neural-network of Watrous and Shastri[8]. In their work, they trained a system to discriminate between suitably processed acoustic signals of the words "no" and "go." Once trained on a single utterance, the system was able to correctly identify other samples of these words from the same speaker. One drawback of their approach was that the weights did not converge to a fixed point. We were therefore particularly interested in whether our system could converge smoothly and rapidly to a stable solution, using the same data, and yet generalize as well as theirs did. This experiment also provided an opportunity to test a solution to the intermediate step training problem.

The architecture was a 16-30-2 network. Each of the input nodes received an input signal corresponding to the energy (sampled every 2.5 milliseconds) as a function of time in one of 16 frequency channels. The input values were normalized to lie in the range 0.0 to 1.0. The values of $d$ and $a$ were 0.9944 and 0.022, respectively. These values were chosen to give a characteristic decay time comparable to the length of each word (they were nearly the same length), and a maximum value computed by the input nodes of 4.0. For an input signal that was part of the word "no", the training signal was (1.0, 0.0), while for the word "go" it was (0.0, 1.0). Thus the outputs that were compared to the training signal can be interpreted as evidence for one word or the other at each time step. The error shown in Fig. 4 is the sum of the squares of the

difference between the desired outputs and the computed outputs for each time step, for both words, after training up to the number of iterations indicated along the x-axis.

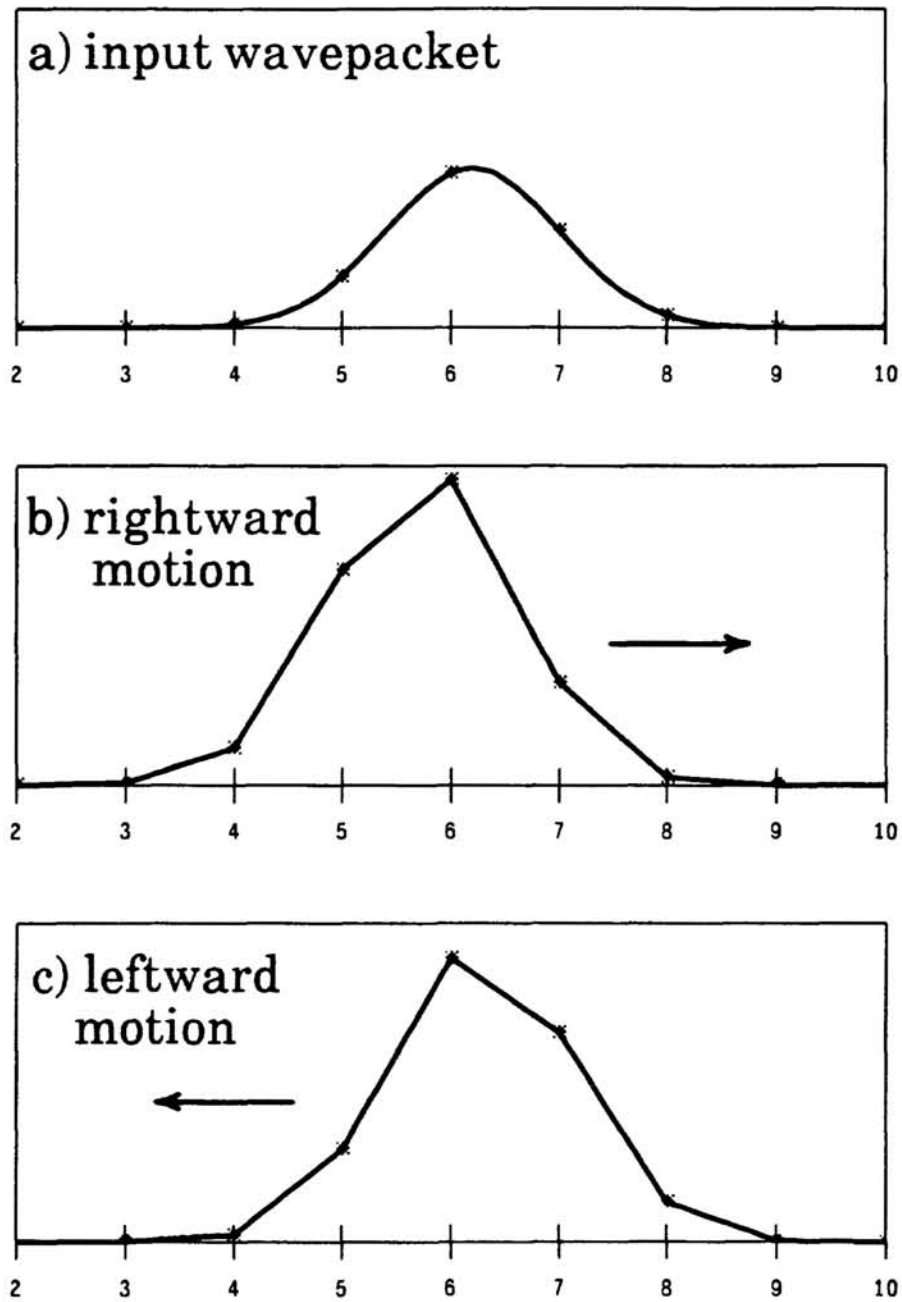

Fig. 2: a) Packet presented to input nodes. The x-axis represents the input nodes. b) Computed values from input nodes during rightward motion. c) Computed values during leftward motion.

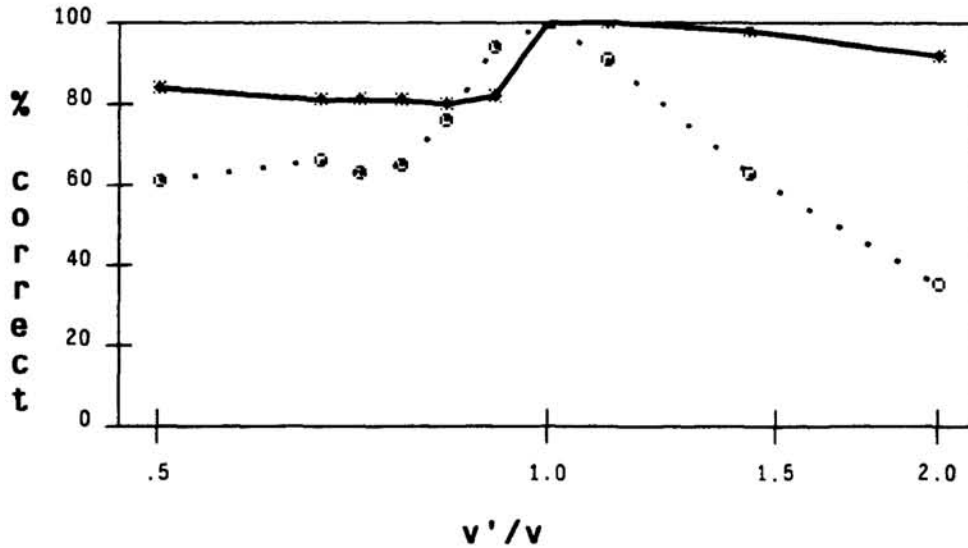

Fig. 3: Performance of motion detection experiment for various velocities. Dashed curve is performance without scaling and solid curve is with the scaling given in Eqs. (2) and (3).

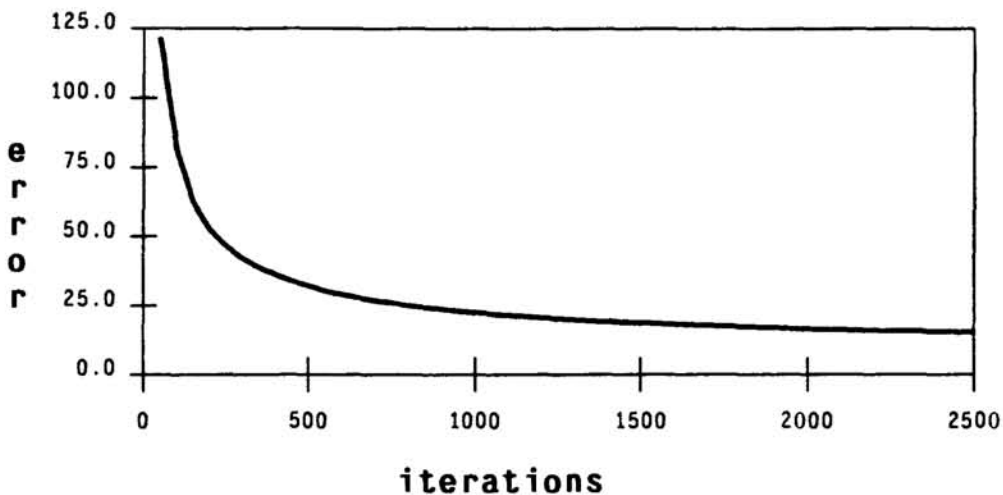

Fig. 4: Error in no/go discrimination as a function of the number of training iterations.

Evidence for each word was obtained by summing the values of the respective nodes over time. This suggests a mechanism for signaling the completion of a sequence: when this sum crosses a certain threshold value, the sequence (in this case, the word) is considered recognized. Moreover, it may be possible to extend this mechanism to apply to the case of connected speech: after a word is recognized, the sums could be reset to zero, and the input nodes reinitialized.

Once we had trained the system on a single utterance, we tested the performance of the resulting weights on additional utterances of the same speaker.

Preliminary results indicate an ability to correctly discriminate between "no" and "go." This suggests that the system has at least a limited ability to generalize in this task domain.

## DISCUSSION

At a more general level, this paper raises and addresses some issues of representation. By choosing input and output representations in a particular way, we are able to make a static optimizer work on a temporal problem while still allowing time to serve as its own representation. In this broader context, one realizes that the choice of capacitive inputs for the input nodes was only one among many possible temporal feature detectors.

Other possibilities include refractory units, derivative units and delayed spike units. Refractory units would compute a value which was some fraction of the current input. The fraction would decrease the more frequently and recently the node had been "on" in the recent past. A derivative unit would have a larger output the more rapidly a signal changed from one time step to the next. A delayed spike unit might have a transfer function of the form $It^n e^{-at}$, where $t$ is the time since the presentation of the signal. This is similar to the function used by Tank and Hopfield[7], but here it could serve a different purpose. The maximum value that a given input generated would be delayed by a certain amount of time. By similarly delaying the training signal, the system could be trained to recognize a given input in the context of signals not only preceding but also following it. An important point to note is that the transfer functions of each of these proposed temporal feature detectors could be rescaled in a manner similar to the capacitive nodes. This would preserve the property of the system that the weights contain information about the temporal sequence to some degree independent of the sampling rate.

An even more ambitious possibility would be to have the system train the parameters, such as $d$ in the capacitive node case. It may be feasible to do this in the same way that weights are trained, namely by taking the partial of the computed error with respect to the parameter in question. Such a system may be able to determine the relevant time scales of a temporal signal and adapt accordingly.

## ACKNOWLEDGEMENTS

We are grateful for fruitful discussions with Jeff Kephart and the help of Raymond Watrous in providing data from his own experiments. This work was partially supported by DARPA ISTO Contract # N00140-86-C-8996 and ONR Contract # N00014-82-0699.

1.  D. Rumelhart, ed., *Parallel Distributed Processing*, (MIT Press, Cambridge, 1986).

2.  J. Denker, ed., *Neural Networks for Computing*, AIP Conf. Proc.,151 (1986).

3.  T. J. Sejnowski and C. R. Rosenberg, *NETtalk: A Parallel Network that Learns to Read Aloud*, Johns Hopkins Univ. Report No. JHU/EECS-86/01 (1986).

4.  J.L. McClelland and J.L. Elman, in *Parallel Distributed Processing*, vol. II, p. 58.

5.  W. Keirstead and B.A. Huberman, *Phys. Rev. Lett.* **56**, 1094 (1986).

6.  A. Lapedes and R. Farber, *Nonlinear Signal Processing Using Neural Networks*, Los Alamos preprint LA-UR-87-2662 (1987).

7.  D. Tank and J. Hopfield, *Proc. Nat. Acad. Sci.*, **84**, 1896 (1987).

8.  R. Watrous and L. Shastri, *Proc. 9th Ann. Conf. Cog. Sci. Soc.*, (Lawrence Erlbaum, Hillsdale, 1987), p. 518.

9.  P. Kanerva, *Self-Propagating Search: A Unified Theory of Memory*, Stanford Univ. Report No. CSLI-84-7 (1984).

10. M.I. Jordan, *Proc. 8th Ann. Conf. Cog. Sci. Soc.*, (Lawrence Erlbaum, Hillsdale, 1986), p. 531.

11. J. Hopfield, *Proc. Nat. Acad. Sci.*, **79**, 2554 (1982).

12. S. Grossberg, *The Adaptive Brain*, vol. II, ch. 6, (North-Holland, Amsterdam, 1987).

13. G. Hinton and T. J. Sejnowski, in *Parallel Distributed Processing*, vol. I, p. 282.

14. B. Gold, in *Neural Networks for Computing*, p. 158.

15. T. Hogg and B.A. Huberman, *Phys. Rev.* **A32**, 2338 (1985).
